# EVOLUTION AND LEARNING IN NEURAL NETWORKS: THE NUMBER AND DISTRIBUTION OF LEARNING TRIALS AFFECT THE RATE OF EVOLUTION

**Ron Keesing** and **David G. Stork***

Ricoh California Research Center    and    *Dept. of Electrical Engineering
2882 Sand Hill Road Suite 115                   Stanford University
Menlo Park, CA 94025                      Stanford, CA 94305
stork@crc.ricoh.com                 stork@psych.stanford.edu

## Abstract

Learning can increase the rate of evolution of a population of biological organisms (the Baldwin effect). Our simulations show that in a population of artificial neural networks solving a pattern recognition problem, no learning or too much learning leads to slow evolution of the genes whereas an intermediate amount is optimal. Moreover, for a given total number of training presentations, fastest evoution occurs if different individuals within each generation receive *different* numbers of presentations, rather than *equal* numbers. Because genetic algorithms (GAs) help avoid local minima in energy functions, our hybrid learning-GA systems can be applied successfully to complex, high-dimensional pattern recognition problems.

## INTRODUCTION

The structure and function of a biological network derives from both its evolutionary precursors and real-time learning. Genes specify (through development) coarse attributes of a neural system, which are then refined based on experience in an environment containing more information — and more *unexpected* information — than the genes alone can represent. Innate neural structure is essential for many high level problems such as scene analysis and language [Chomsky, 1957].

Although the Central Dogma of molecular genetics [Crick, 1970] implies that information *learned* cannot be *directly* transcribed to the genes, such information *can* appear in the genes through an indirect Darwinian process (see below). As such, learning can change the rate of evolution — the Baldwin effect [Baldwin, 1896]. Hinton and Nowlan [1987] considered a closely related process in artificial neural networks, though they used stochastic search and not learning *per se*. We present here analyses and simulations of a hybrid evolutionary-learning system which uses gradient-descent learning as well as a genetic algorithm, to determine network connections.

Consider a population of networks for pattern recognition, where initial synaptic weights (weights "at birth") are determined by genes. Figure 1 shows the Darwinian fitness of networks (i.e., how many patterns each can correctly classify) as a function the weights. Iso-fitness contours are *not* concentric, in general. The tails of the arrows represent the synaptic weights of networks at birth. In the case of evolution *without* learning, network **B** has a higher fitness than does **A**, and thus would be preferentially selected. In the case of gradient-descent learning before selection, however, network **A** has a higher after-learning fitness, and would be preferentially selected (tips of arrows). Thus learning can change which individuals will be selected and reproduce, in particular favoring a network (here, **A**) whose genome is "good" (i.e., initial weights "close" to the optimal), despite its poor performance at birth. Over many generations, the choice of "better" genes for reproduction leads to new networks which require less learning to solve the problem — they are closer to the optimal. The rate of gene evolution is increased by learning (the Baldwin effect).

### Iso-fitness contours

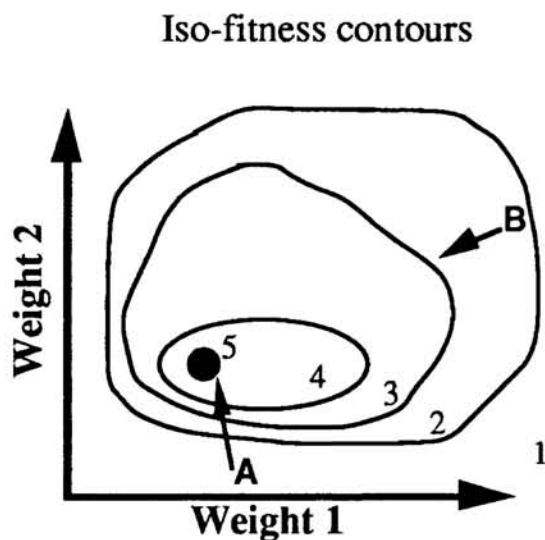

**Weight 2**

**Weight 1**

**Figure 1:** Iso-fitness contours in synaptic weight space. The black region corresponds to perfect classifications (fitness = 5). The weights of two networks are shown *at birth* (tails of arrows), and *after learning* (tips of arrows). At birth, **B** has a higher fitness score (2) than does **A** (1); a pure genetic algorithm (without learning) would preferentially reproduce **B**. *With* learning, though, **A** has a higher fitness score (4) than **B** (2), and would thus be preferentially reproduced. Since **A**'s genes are "better" than **B**'s, learning can lead to selection of better genes.

Surprisingly, *too much* learning leads to *slow* evolution of the genome, since after sufficient training in each generation, *all* networks can perform perfectly on the pattern recognition task, and thus are equally likely to pass on their genes, regardless of whether they are "good" or "bad." In Figure 1, if both **A** and **B** continue learning, eventually both will identify all five patterns correctly. **B** will be just as likely to reproduce as **A**, even though **A**'s genes are "better." Thus the rate of evolution will be decreased — too much learning is worse than an intermediate amount — or even no — learning.

## SIMULATION APPROACH

Our system consists of a population of 200 networks, each for classifying pixel images of the first five letters of the alphabet. The 9 x 9 input grid is connected to four 7 x 7 sets of overlapping 3 x 3 orientation detectors; each detector is fully connected by modifiable weights to an output layer containing five category units (Fig. 2).

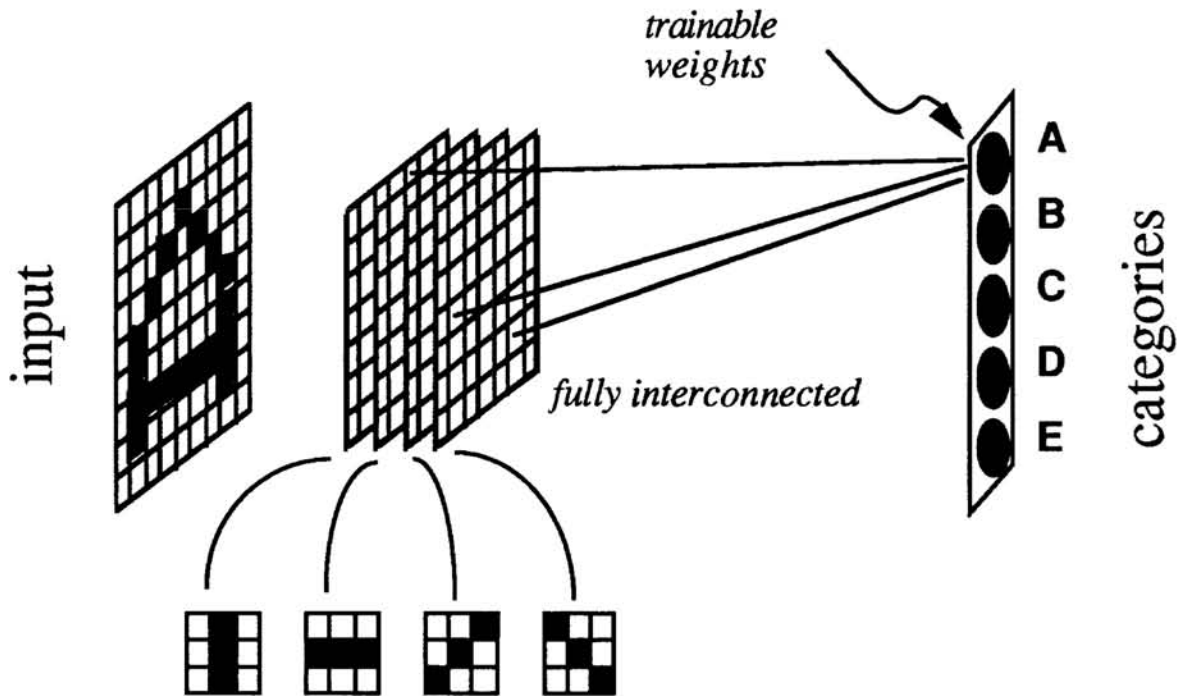

**Figure 2**: Individual network architecture. The 9x9 pixel input is detected by each of four orientation selective input layers (7x7 unit arrays), which are fully connected by trainable weights to the five category units. The network is thus a simple perceptron with 196 (=4x7x7) inputs and 5 outputs. Genes specify the initial connection strengths.

Each network has a 490-bit gene specifying the initial weights (Figure 3). For each of the 49 filter positions and 5 categories, the gene has two bits

which specify *which* orientation is initially most strongly connected to the category unit (by an arbitrarily chosen factor of 3:1). During training, the weights from the filters to the output layer are changed by (supervised) perceptron learning. Darwinian fitness is given by the number of patterns correctly classified *after* training. We use fitness-proportional reproduction and the standard genetic algorithm processes of replication, mutation, and cross-over [Holland, 1975]. Note that while fitness may be measured after training, reproduction is of the genes present at birth, in accord with the Central Dogma. This is not a Lamarkian process.

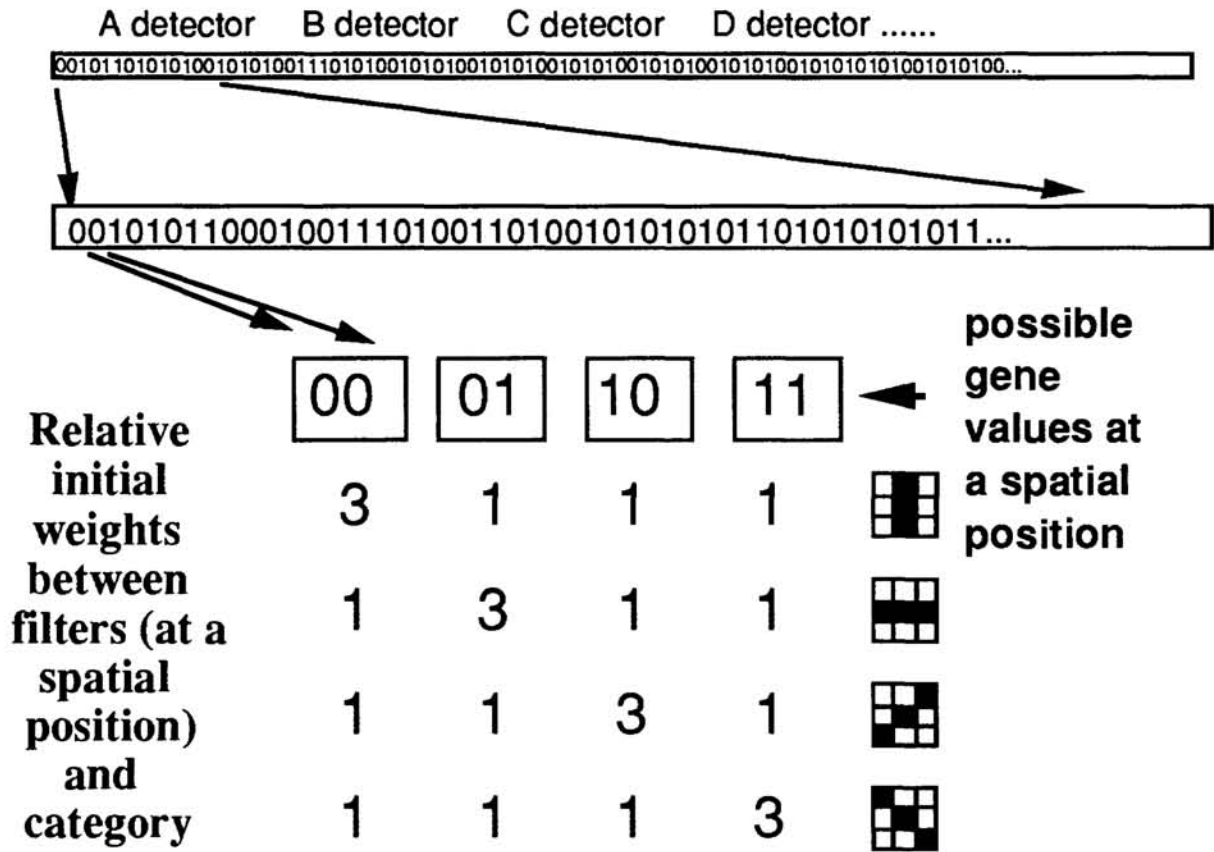

**Figure 3:** The genetic representation of a network. For each of the five category units, 49 two-bit numbers describe which of the four orientation units is most strongly connected at each position within the 7x7 grid. This unit is given a relative connection strength of 3, while the other three orientation units at that position are given a relative strength of 1.

For a given total number of teaching presentations, reproductive fitness might be defined in many ways, including categorization score at the end of learning or during learning; such functions will lead to different rates of evolution. We show simulations for two schemes: in *uniform learning* each network received the same number (e.g., 20) of training presentations; in

*distributed learning* networks received a randomly chosen number (10, 34, 36, 16, etc.) of presentations.

## RESULTS AND DISCUSSION

Figure 4 shows the population average fitness *at birth*. The lower curve shows the performance of the genetic algorithm alone; the two upper curves represent genetypic evolution — the amount of information within the genes — when the genetic algorithm is combined with gradient-descent learning. Learning increases the rate of evolution — both uniform and distributed learning are significantly better than no learning. The fitness *after* learning in a generation (not shown) is typically only 5% higher than the fitness at birth. Such a small improvement at a *single* generation cannot account for the overall high performance at later generations. A network's performance — even after learning — is more dependent upon its *ancestors* having learned than upon *its* having learned the task.

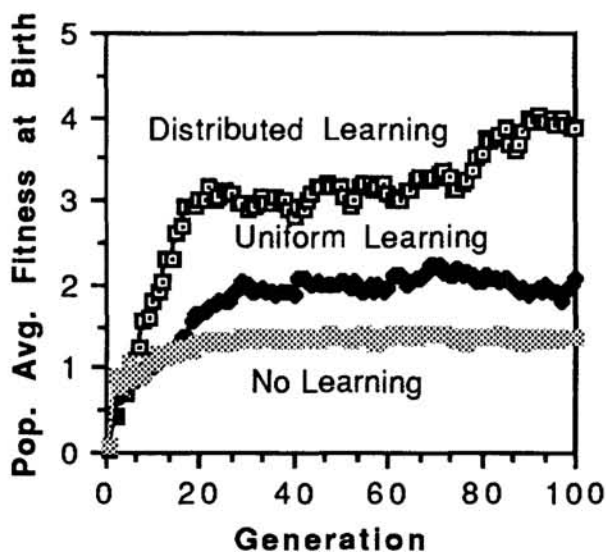

**Pop. Avg. Fitness at Birth for Different Learning Schemes**

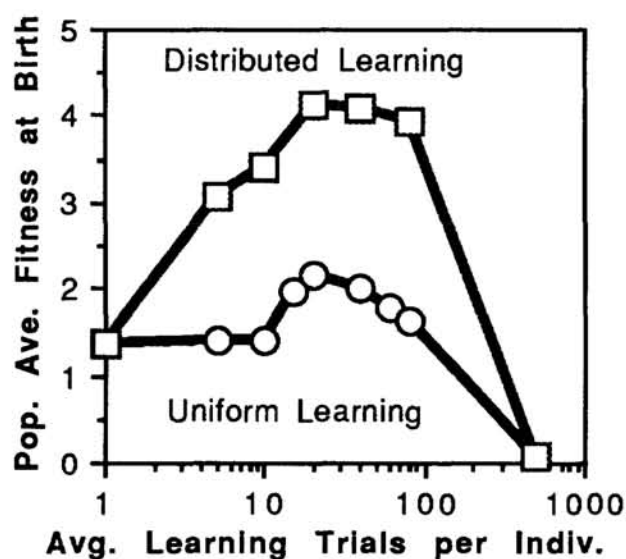

**Ave. Fitness at Generation 100 Depends on Amount of Training**

**Figure 4:** Learning guides the rate of evolution. In uniform learning, every network in every generation receives 20 learning presentations; in the distributed learning scheme, any network receives a number of patterns randomly chosen between 0 and 40 presentations (mean = 20). Clearly, evolution with learning leads to superior genes (fitness *at birth*) than evolution without learning.

**Figure 5:** Selectivity of learning-evolution interactions. Too little or too much learning leads to slow evolution (population fitness at birth at generation 100) while an intermediate amount of learning leads to significantly higher such fitness. This effect is significant in both learning schemes. (Each point represents the mean of five simulation runs.)

Figure 5 illustrates the tuning of these learning-evolution interactions, as discussed above: too little or too much learning leads to poorer evolution than does an intermediate amount of learning. Given excessive learning (e.g., 500 presentations) *all* networks perform perfectly. This leads to the *slowest* evolution, since selection is independent of the quality of the genes.

Note too in Fig. 4 that distributed learning leads to significantly faster evolution (higher fitness at any particular generation) than uniform learning. In the uniform learning scheme, once networks have evolved to a point in weight space where they (and their offspring) can identify a pattern after learning, there is no more "pressure" on the genes to evolve. In Figure 6, both **A** and **B** are able to identify three patterns correctly after uniform learning, and hence both will reproduce equally. However, in the distributed learning scheme, one of the networks may (randomly) receive a small amount of learning. In such cases, **A**'s reproductive fitness will be unaffected, because it is able to solve the patterns without learning, while **B**'s fitness will decrease significantly. Thus in the distributed learning scheme (and in schemes in which fitness is determined in part *during* learning), there is "pressure" on the genes to improve at every generation. Diversity is, a driving force for evolution. Our distributed learning scheme leads to a greater diversity of fitness throughout a population.

## Iso-fitness contours

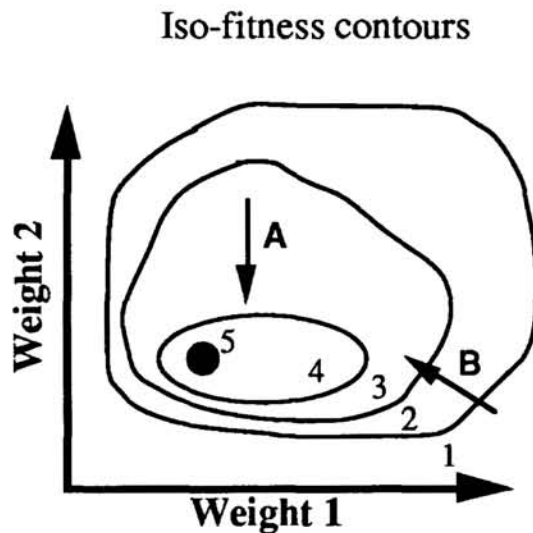

**Figure 6:** Distributed learning leads to faster evolution than uniform learning. In uniform learning, (shown above) **A** and **B** have equal reproductive fitness, even though **A** has "better" genes. In distributed learning, **A** will be more likely to reproduce when it (randomly) receives a small amount of learning (shorter arrow) than **B** will under similar circumstances. Thus "better" genes will be more likely to reproduce, leading to faster evolution.

## CONCLUSIONS

Evolutionary search via genetic algorithms is a powerful technique for avoiding local minima in complicated energy landscapes [Goldberg, 1989; Peterson, 1990], but is often slow to converge in large problems. Conventional genetic approaches consider only the reproductive fitness of

the genes; the slope of the fitness landscape in the immediate vicinity of the genes is ignored.  Our hybrid evolutionary-learning approach utilizes the gradient of the local fitness landscape, along with the fitness of the genes, in determining survival and reproduction.

We have shown that this technique offers advantages over evolutionary search alone in the single-minimum landscape given by perceptron learning. In a simple pattern recognition problem, the hybrid system performs twice as well as a genetic algorithm alone.  A hybrid system with distributed learning, which increases the "pressure" on the genes to evolve at every generation, performs four times as well as a genetic algorithm.  In addition, we have demonstrated that there exists an optimal average amount of learning in order to increase the rate of evolution — too little or too much learning leads to slower evolution.  In the extreme case of too much learning, where all networks are trained to perfect performance, there is no improvement of the genes.  The advantages of the hybrid approach in landscapes with *multiple* minima can be even more pronounced [Stork and Keesing, 1991].

## Acknowledgments
Thanks to David Rumelhart, Marcus Feldman, and Aviv Bergman for useful discussions.

## References
Baldwin, J. M. "A new factor in evolution," *American Naturalist* **30**, 441-451 (1896)

Chomsky, N. *Syntactic Structures* The Hague:  Mouton (1957)

Crick, F. W. "Central Dogma of Molecular Biology," *Nature* **227**, 561-563 (1970)

Goldberg, D. E. *Genetic Algorithms in Search, Optimization & Machine Learning* Reading, MA:  Addison-Wesley (1989).

Hinton, G. E. and Nowlan, S. J. "How learning can guide evolution," *Complex Systems* **1**, 495-502 (1987)

Holland, J. H. *Adaptation in Natural and Artificial Systems* University of Michigan Press (1975)

Peterson, C. "Parallel Distributed Approaches to Combinatorial Optimization: Benchmanrk Studies on Traveling Salesman Problem," *Neural Computation* **2**, 261-269 (1990).

Stork, D. G. and Keesing, R. "The distribution of learning trials affects evolution in neural networks" (1991, submitted).